# 3D Object Recognition:
# A Model of View-Tuned Neurons

**Emanuela Bricolo**     **Tomaso Poggio**
Department of Brain and Cognitive Sciences
Massachusetts Institute of Technology
Cambridge, MA 02139
{emanuela,tp}@ai.mit.edu

**Nikos Logothetis**
Baylor College of Medicine
Houston, TX 77030
nikos@bcmvision.bcm.tmc.edu

## Abstract

In 1990 Poggio and Edelman proposed a view-based model of object recognition that accounts for several psychophysical properties of certain recognition tasks. The model predicted the existence of view-tuned and view-invariant units, that were later found by Logothetis et al. (Logothetis et al., 1995) in IT cortex of monkeys trained with views of specific paperclip objects. The model, however, does not specify the inputs to the view-tuned units and their internal organization. In this paper we propose a model of these view-tuned units that is consistent with physiological data from single cell responses.

## 1   INTRODUCTION

Recognition of specific objects, such as recognition of a particular face, can be based on representations that are object centered, such as 3D structural models. Alternatively, a 3D object may be represented for the purpose of recognition in terms of a set of views. This latter class of models is biologically attractive because model acquisition – the learning phase – is simpler and more natural.

A simple model for this strategy of object recognition was proposed by Poggio and Edelman (Poggio and Edelman, 1990). They showed that, with few views of an object used as training examples, a classification network, such as a Gaussian radial basis function network, can learn to recognize novel views of that object, in partic-

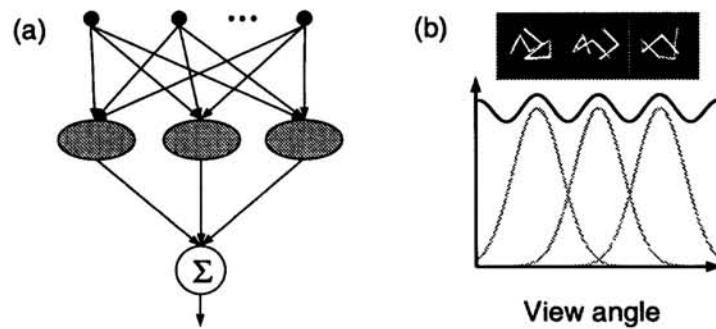

Figure 1: (a) Schematic representation of the architecture of the Poggio-Edelman model. The shaded circles correspond to the view-tuned units, each tuned to a view of the object, while the open circle correspond to the view-invariant, object specific output unit. (b) Tuning curves of the view-tuned (gray) and view-invariant (black) units.

ular views obtained by in depth rotation of the object (translation, rotation in the image plane and scale are probably taken care by object independent mechanisms). The model, sketched in Figure 1, makes several prediction about limited generalization from a single training view and about characteristic generalization patterns between two or more training views (Bülthoff and Edelman, 1992). Psychophysical and neurophysiological results support the main features and predictions of this simple model. For instance, in the case of novel objects, it has been shown that when subjects –both humans and monkeys– are asked to learn an object from a single unoccluded view, their performance decays as they are tested on views farther away from the learned one (Bülthoff and Edelman, 1992; Tarr and Pinker, 1991; Logothetis et al., 1994). Additional work has shown that even when 3D information is provided during training and testing, subjects recognize in a view dependent way and cannot generalize beyond 40 degrees from a single training view (Sinha and Poggio, 1994).

Even more significantly, recent recordings in inferotemporal cortex (IT) of monkeys performing a similar recognition task with paperclip and amoeba-like objects, revealed cells tuned to specific views of the learned object (Logothetis et al., 1995). The tuning, an example of which is shown in Figure 3, was presumably acquired as an effect of the training to views of the particular object. Thus an object can be thought as represented by a set of cells tuned to several of its views, consistently with finding of others (Wachsmuth et al., 1994). This simple model can be extended to deal with symmetric objects (Poggio and Vetter, 1992) as well as objects which are members of a *nice* class (Vetter et al., 1995): in both cases generalization from a single view may be significantly greater than for objects such as the paperclips used in the psychophysical and physiological experiments.

The original model of Poggio and Edelman has a major weakness: it does not specify which features are inputs to the view-tuned units and what is stored as a representation of a view in each unit. The simulation data they presented employed features such as the x,y coordinates of the object vertices in the image plane or the angles between successive segments. This representation, however, is biologically implausible and specific for objects that have easily detectable vertices and measurable angles, like paperclips. In this paper, we suggest a view representation which is more biologically plausible and applies to a wider variety of cases. We will also show that this extension of the Poggio-Edelman model leads to properties that are

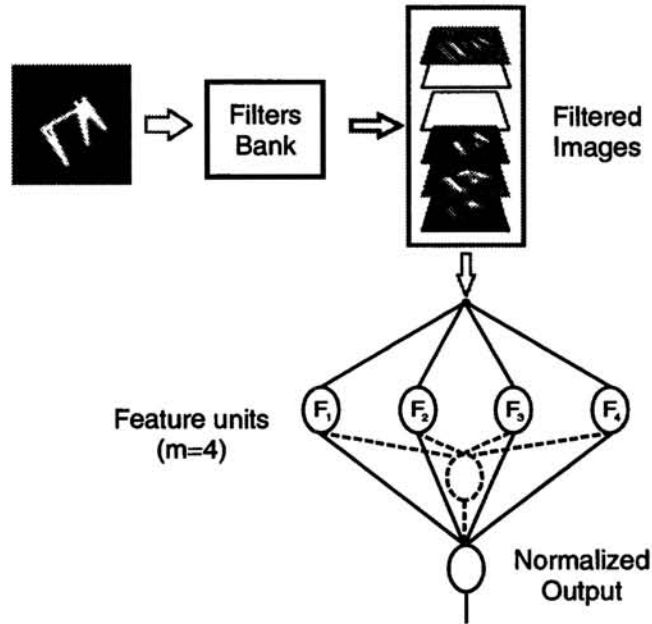

Figure 2: Model overview: during the training phase the images are first filtered through a bank of steerable filters. Then a number of image locations are chosen by an attentional mechanism and the vector of filtered values at these locations is stored in the feature units.

consistent with the cell response to the same objects.

## 2   A MODEL OF VIEW-TUNED UNITS

Our approach consists in representing a view in terms of a few local features, which can be regarded as local configurations of grey-levels. Suppose one point in the image of the object is chosen. A feature vector is computed by filtering the image with a set of filters with small support centered at the chosen location. The vector of filter responses serves as a description of the local pattern in the image. Four such points were chosen, for example, in the image of Figure 3a, where the white squares indicate the support of the bank of filters that were used. Since the support is local but finite, the value of each filter depends on the pattern contained in the support and not only on the center pixel; since there are several filters one expects that the vector of values may uniquely represent the local feature, for instance a corner of the paperclip.

We used filters that are somewhat similar to oriented receptive fields in V1 (though it is far from being clear whether some V1 cells behave as linear filters). The ten filters we used are the same steerable filters (Freeman and Adelson, 1991) suggested by Ballard and Rao (Rao and Ballard, 1995; Leung et al., 1995). The filters were chosen to be a basis of steerable 2-dimensional filters up to the third order. If $G_n$ represents the $n$th derivative of a Gaussian in the $x$ direction and we define the rotation operator $(...)^\theta$ as the operator that rotates a function through an angle $\theta$ about the origin, the ten basis filters are:

$$G_n^{\theta_k} \quad \begin{array}{l} n = 0, 1, 2, 3 \\ \theta_k = 0, ..., k\pi/(n+1), k = 1, ...n \end{array} \tag{1}$$

Therefore for each one of the chosen locations $m$ in the image we have a 10-value array $\mathbf{T}_m$ given by the output of the filters bank.

$$\mathbf{T}_m = ((I * G_0^0)|_m, (I * G_1^0)|_m, (I * G_1^{\pi/2})|_m, ..., (I * G_3^{3\pi/4})|_m) \qquad (2)$$

The representation of a given view of an object is then the following. First $m = 1, ..., M$ locations are chosen, then for each of these $M$ locations the 10-valued vectors $\mathbf{T}_m$ are computed and stored. These $M$ vectors, with $M$ typically between 1 and 4, form the representation of the view which is learned and commited to memory.

How are the locations chosen? Precise location is not critical. Feature locations can be chosen almost randomly. Of course each specific choice will influence properties of the unit but precise location does not affect the qualitative properties of the model, as verified in simulation experiments. Intuitively, features should be centered at salient locations in the object where there are large changes in contrast and curvature. We have implemented (Riesenhuber and Bricolo, in preparation) a simple attentional mechanism that chooses locations close to edges with various orientations[1]. The locations shown in Figures 3 and 4 were obtained with this unsupervised technique. We emphasize however that all the results and conclusions of this paper do not depend on the specific location of the feature or the precise procedure used to choose them.

We have described so far the learning phase and how views are represented and stored. When a new image $\mathbf{V}$ is presented, recognition takes place in the following way. First the new image is filtered through the bank of filters. Thus at each pixel location $i$ we have the vector of values $\mathbf{f}_i$ provided by the filters. Now consider the first stored vector $\mathbf{T}_1$. The closest $\mathbf{f}_i^*$ is found searching over all $i$ locations and the distance $D_1 = \|\mathbf{T}_1 - \mathbf{f}_i^*\|$ is computed. This process is repeated for the other feature vectors $\mathbf{T}_m$ for $m = 2, ..., M$. Thus for the new image $\mathbf{V}$, $M$ distances $D_m$ are computed; the distance $D_m$ is therefore the distance to the stored feature $\mathbf{T}_m$ of the closest image vector searched over the whole image.

The model uses these $M$ distances as exponents in $M$ Gaussian units. The output of the system is a weighted average of the output of these units with an output non linearity of the sigmoidal type:

$$\mathbf{Y_V} = h\left(\sum_{m=1}^{M} c_m e^{-\frac{D_m^2}{2\sigma^2}}\right) \qquad (3)$$

In the simulations presented in this paper we estimated $\sigma$ from the distribution of distances over several images; the $c_m$ are $c_m = M^{-1}$, since we have only one training view; h is $h(x) = 1/(1 - e^{-x})$.

In Figure 3b we see the result obtained by simply combining linearly the output of the four feature detectors followed by the sigmoidal non linearity (Figure 4a shows another example). We have also experimented with a multiplicative combination of the output of the feature units. In this case the system performs an AND of the $M$ features. If the response to the distractors is used to set a threshold for

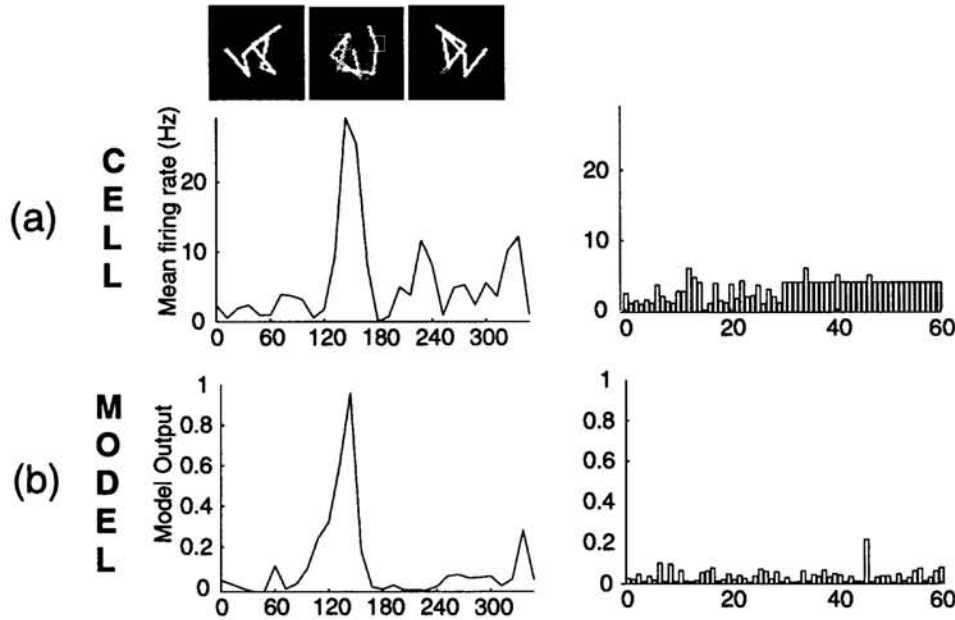

Figure 3: Comparison between a model view-tuned unit and cortical neuron tuned to a view of the same object. (a) Mean spike rate of an inferotemporal cortex cell recorded in response to views of a specific paperclip [left] and to a set of 60 distractor paperclip objects [right],(Logothetis and Pauls, personal communication). (b) Model response for the same set of objects. This is representative for other cells we have simulated, thought there is considerable variability in the cells (and the model) tuning.

classification, then the two versions of the system behave in a similar way. Similar results (not shown) were obtained using other kinds of objects.

## 3  RESULTS

### 3.1  COMPARISON BETWEEN VIEW-TUNED UNITS AND CORTICAL NEURONS

Electrophysiological investigations in alert monkeys, trained to recognize wire-like objects presented from any view show that the discharge rate of many IT neurons is a bell-shaped function of orientation centered on a preferred view (Logothetis et al., 1995). The properties of the units here described are comparable to those of the cortical neurons (see Figure 3). The model was tested with the exactly the same objects used in the physiological experiments. As training view for the model we used the view preferred by the cell (the cell became tuned presumably as an effect of training during which the monkey was shown in this particular case several views of this object).

### 3.2  OCCLUSION EXPERIMENTS

What physiological experiments could lend additional support to our model? A natural question concerns the behavior of the cells when various parts of the object are occluded. The predictions of our model are given in Figure 4 for a specific object and a specific choice of feature units ($m = 4$) and locations.

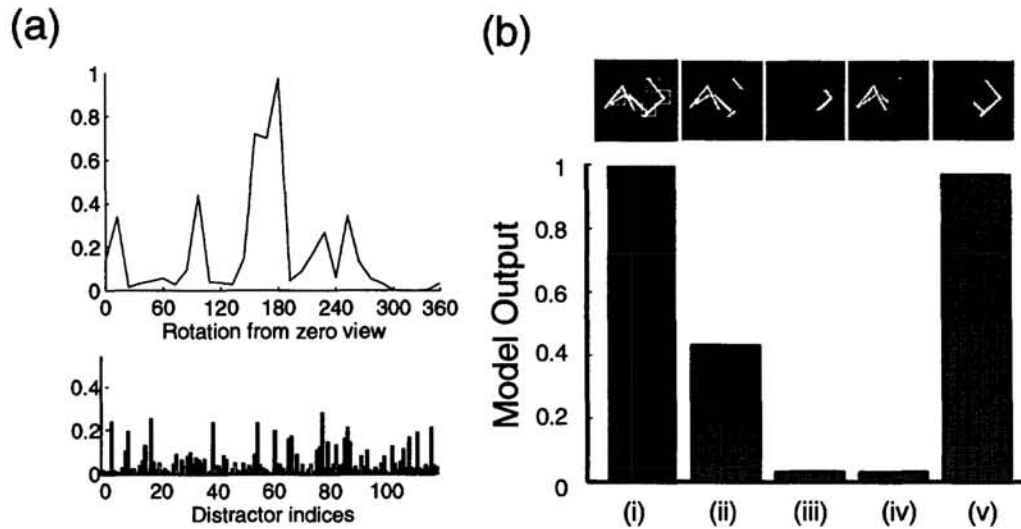

Figure 4: (a) Model behavior in response to a learned object in full view (highlighted on the learned image are the positions of the four features) at different rotations and to 120 other paperclip objects (distractors), (b) Response dependence on occluder characteristics: (i) object in full view at learned location, (ii) object occluded with a small occluder, (iii) occluded region in (ii) presented in isolation, (iv-v) same as (ii-iii) but with a larger occluder.

The simulations show that the behavior depends on the position of key features with respect to the occluder itself. Occluding a part of the object can drastically reduce the response to that specific view (Figure 4b(ii-iv)) because of interference with more than one feature. But since the occluded region does not completely overlap with the occluded features (considering the support of the filters), the presentation of this region alone does not always evoke a significant response (Figure 4b(iii-v)).

## 4   Discussion

Poggio and Edelman model was designed specifically for paperclip objects and did not explicitly specify how to compute the response for any object and image. In this paper we fill this gap and propose a model of these IT cells that become view tuned as an effect of training. The key aspect of the model is that it relies on a few local features (1-4) that are computed and stored during the training phase. Each feature is represented as the set of responses of oriented filters at one location in the image. During recognition the system computes a robust conjunction of the best matches to the stored features.

Clearly, the version of the model described here does not exploit information about the geometric configuration of the features. This information is available once the features are detected and can be critical to perform more robust recognition. We have devised a model of how to use the relative position of the features $f_i^*$ in the image. The model can be made translation and scale invariant in a biologically plausible way by using a network of cells with linear receptive fields, similar in spirit to a model proposed for spatial representation in the parietal cortex (Pouget and Sejnowski, 1996). Interestingly enough, this additional information is not needed to account for the selectivity and the generalization properties of the IT cells we

have considered so far. The implication is that IT cells may not be sensitive to the overall configuration of the stimulus but to the presence of moderately complex local features (according to our simulations, the number of necessary local features is greater than one for the most selective neurons, such as the one of Figure 3a). Scrambling the image of the object should therefore preserve the selectivity of the neurons, *provided* this can be done without affecting the filtering stage. In practice this may be very difficult. Though our model is still far from being a reasonable neuronal model, it can already be used to make useful predictions for physiological experiments which are presently underway.

## Footnotes

[1] A saliency map is at first constructed as the average of the convolutions of the image with four directional filters (first order steerable filters with $\theta = 0, ..., k\pi/(4), k = 1, ...4$). The locations with higher saliency are extracted one at the time. After each selection, a region around the selected position is inhibited to avoid selecting the same feature over again.

## References

Bülthoff, H. and Edelman, S. (1992). Psychophisical support for a two-dimensional view interpolation theory of object recognition. *Proceedings of the National Academy of Science. USA*, 89:60–64.

Freeman, W. and Adelson, E. (1991). The design and use of steerable filters. *IEEE transactions on Pattern Analysis and Machine Intelligence*, 13(9):891–906.

Leung, T., Burl, M., and Perona, P. (1995). Finding faces in cluttered scenes using random labelled graph matching. In *Proceedings of the 5th Internatinal Conference on Computer Vision*, Cambridge, Ma.

Logothetis, N., Pauls, J., Bülthoff, H., and Poggio, T. (1994). View dependent object recognition by monkeys. *Current Biology*, 4(5):401–414.

Logothetis, N., Pauls, J., and Poggio, T. (1995). Shape representation in the inferior temporal cortex of monkeys. *Current Biology*, 5(5):552–563.

Poggio, T. and Edelman, S. (1990). A network that learns to recognize three-dimensional objects. *Nature*, 343:263–266.

Poggio, T. and Vetter, T. (1992). Recognition and structure from one 2d model view: observations on prototypes, object classes and symmetries. Technical Report A.I. Memo No.1347, Massachusetts Institute of Technnology, Cambridge, Ma.

Pouget, A. and Sejnowski, T. (1996). Spatial representations in the parietal cortex may use basis functions. In Tesauro, G., Touretzky, D., and Leen, T., editors, *Advances in Neural Information Processing Systems*, volume 7, pages 157–164. MIT Press.

Rao, R. and Ballard, D. (1995). An active vision architecture based on iconic representations. *Artificial Intelligence Journal*, 78:461–505.

Sinha, P. and Poggio, T. (1994). View-based strategies for 3d object recognition. Technical Report A.I. Memo No.1518, Massachusetts Institute of Technnology, Cambridge, Ma.

Tarr, M. and Pinker, S. (1991). Orientation-dependent mechanisms in shape recognition: further issues. *Psychological Science*, 2(3):207–209.

Vetter, T., Hurlbert, A., and Poggio, T. (1995). View-based models of 3d object recognition: Invariance to imaging transformations. *Cerebral Cortex*, 3(261–269).

Wachsmuth, E., Oram, M., and Perrett, D. (1994). Recognition of objects and their component parts: Responses of single units in the temporal cortex of the macaque. *Cerebral Cortex*, 4(5):509–522.
